# Optimal context separation of spiking haptic signals by second-order somatosensory neurons

**Romain Brasselet**
CNRS - UPMC Univ Paris 6, UMR 7102
F 75005, Paris, France
romain.brasselet@upmc.fr

**Roland S. Johansson**
UMEA Univ, Dept Integr Medical Biology
SE-901 87 Umea, Sweden
roland.s.johansson@physiol.umu.se

**Angelo Arleo**
CNRS - UPMC Univ Paris 6, UMR 7102
F 75005, Paris, France
angelo.arleo@upmc.fr

## Abstract

We study an encoding/decoding mechanism accounting for the relative spike timing of the signals propagating from peripheral nerve fibers to second-order somatosensory neurons in the cuneate nucleus (CN). The CN is modeled as a population of spiking neurons receiving as inputs the spatiotemporal responses of real mechanoreceptors obtained via microneurography recordings in humans. The efficiency of the haptic discrimination process is quantified by a novel definition of entropy that takes into full account the metrical properties of the spike train space. This measure proves to be a suitable decoding scheme for generalizing the classical Shannon entropy to spike-based neural codes. It permits an assessment of neurotransmission in the presence of a large output space (i.e. hundreds of spike trains) with 1 ms temporal precision. It is shown that the CN population code performs a complete discrimination of 81 distinct stimuli already within 35 ms of the first afferent spike, whereas a partial discrimination (80% of the maximum information transmission) is possible as rapidly as 15 ms. This study suggests that the CN may not constitute a mere synaptic relay along the somatosensory pathway but, rather, it may convey optimal contextual accounts (in terms of fast and reliable information transfer) of peripheral tactile inputs to downstream structures of the central nervous system.

## 1  Introduction

During haptic exploration tasks, forces are applied to the skin of the hand, and in particular to the fingertips, which constitute the most sensitive parts of the hand and are prominently involved in object manipulation/recognition tasks. Due to the visco-elastic properties of the skin, forces applied to the fingertips generate complex non-linear deformation dynamics, which makes it difficult to predict how these forces can be transduced into percepts by the somatosensory system. Mechanoreceptors innervate the epidermis and respond to the mechanical indentations and deformations of the skin. They send direct projections to the spinal cord and to the cuneate nucleus (CN), which constitutes an important synaptic relay of the ascending somatosensory pathway. The CN projects to several areas of the central nervous system (CNS), including the cerebellum and the thalamic ventrolateral posterior nucleus, which in turn projects to the primary somatosensory cortex. The main objective of this study is to investigate the role of the CN in mediating optimal feed-forward encoding/decoding of somatosensory information.

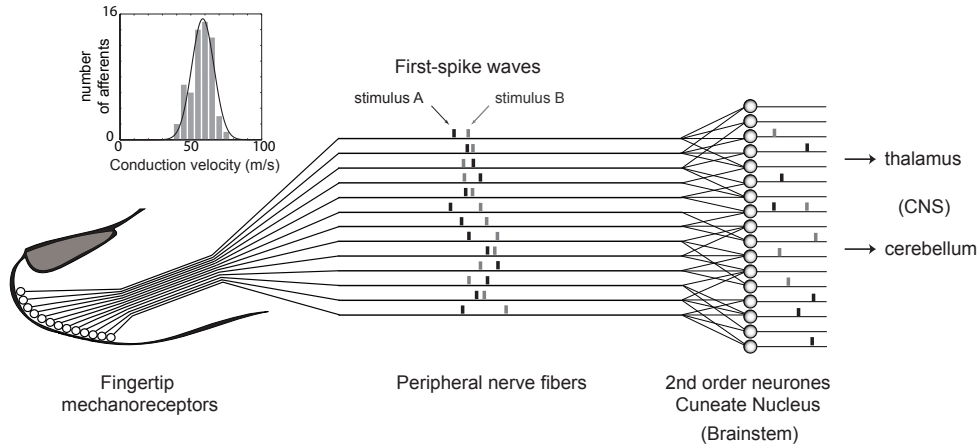

Figure 1: Overview of the ascending pathway from primary tactile receptors of the fingertip to $2^{nd}$ order somatosensory neurons in the cuneate nucleus of the brainstem.

Recent microneurography studies in humans [9] suggest that the relative timing of impulses from ensembles of mechanoreceptor afferents can convey information about contact parameters faster than the fastest possible rate code, and fast enough to account for the use of tactile signals in natural manipulation. Even under the most favorable conditions, discrimination based on firing rates takes on average 15 to 20 ms longer than discrimination based on first spike latency [9, 10]. Estimates of how early the sequence in which afferents are recruited conveys information needed for the discrimination of contact parameters indicate that, among mechanoreceptors, the FA-I population provides the fastest reliable discrimination of both surface curvature and force direction. Reliable discrimination can take place after as few as some five FA-I afferents are recruited, which can occur a few milliseconds after the first impulse in the population response [10].

Encoding and decoding of sensory information based on the timing of neural discharges, rather than (or in addition to) their rate, has received increasing attention in the past decade [7, 22]. In particular, the high information content in the timing of the first spikes in ensembles of central neurons has been emphasized in several sensory modalities, including the auditory [3, 16], visual [4, 6], and somatosensory [17] systems. If relative spike timing is fundamental for rapid encoding and transfer of tactile events in manipulation, then how do neurons read out information carried by a temporal code? Various decoding schemes have been proposed to discriminate between different spatiotemporal sequences of incoming spike patterns [8, 13, 1, 7].

Here, we investigate an encoding/decoding mechanism accounting for the relative spike timing of signals propagating from primary tactile afferents to $2^{nd}$ order neurons in the CN (Fig. 1). The population coding properties of a model CN network are studied by employing as peripheral signals the responses of real mechanoreceptors obtained via microneurography recordings in humans. We focus on the first spike of each mechanoreceptor, according to the hypothesis that the variability in the first-spike latency domain with respect to stimulus feature (e.g. the direction of the force) is larger than the variability within repetitions of the same stimulus [9]. Thus, each tactile stimulus consists of a single volley of spikes (black and gray waves in Fig. 1) forming a spatiotemporal response pattern defined by the first-spike latencies across the afferent population (Fig. S1).

## 2 Methods

### 2.1 Human microneurography data

In order to investigate fast encoding/decoding mechanisms of haptic signals, we concentrate on the responses of FA-I mechanoreceptors only [9]. The stimulus state space is defined according to a set of four primary contact parameters:

- the curvature of the probe ($C = \{0, 100, 200\} \ m^{-1}, |C| = 3$),
- the magnitude of the applied force ($F = \{1, 2, 4\}N, |F| = 3$),
- the direction of the force ($O = \{$Ulnar, Radial, Distal, Proximal, Normal$\}, |D| = 5$),
- the angle of the force relative to the normal direction ($A = \{5, 10, 20\}°, |A| = 3$).

In total, we consider the responses of 42 FA-I mechanoreceptors to 81 distinct stimuli. The propagation velocity distribution across the set of primary projections onto $2^{nd}$ order CN neurons is considered by fitting experimental observations [11, 21] (see Fig. 1, upper-left inset). Each primary afferent is assigned a conduction speed equal to the mean of the experimental distribution. An average peripheral nerve length of $1 \ m$ (from the fingertip to the CN) is then taken to compute the corresponding conduction delay.

## 2.2 Cuneate nucleus model and synaptic plasticity rule

Single unit discharges at the CN level are modeled according to the spike-response model (SRM) [5] (see Supporting Material Sec. A.1). The parameters determining the response of the CN single neuron model are set according to *in vivo* electrophysiological recordings by H. Jörntell (unpublished data). Fig. 2A shows a sample firing pattern that illustrates the spike timing reliability property [14] of the model CN neuron. We assume that the stochasticity governing the entire mechanoreceptors-CN pathway can be represented by the probability function that determines the electro-responsiveness properties of the SRM.

The CN network is modeled as a population of SRM units. The connectivity layout of the mechanoreceptor-to-CN projections is based on neuroanatomical data [12], which suggests an average divergence/convergence ratio of 1700/300. This asymmetric coupling is in favor of a fast feed-forward encoding/decoding process occurring at the CN network level. Based on this divergence/convergence data, and given that there are around 2000 mechanoreceptors at each fingertip (and that the CN is somatotopically organized at least to the precision of the finger), there must exist around 12000 CN neurons coding for the tactile information coming from each fingertip. These data suggest a probability of connection between a mechanoreceptor and a CN cell of 0.15. In order to test the hypothesis of a purely feed-forward information transfer at the CN level, no collateral projections between CN neurons are considered in the current version of the model.

We put forth the hypothesis that the efficacy of the mechanoreceptor-CN synapses is regulated according to spike-timing-dependent plasticity (STDP, [1, 15]). In particular, we employ a STDP rule specifically developed for the SRM [20]. This learning rule optimizes the information transmission property of a single SRM neuron, accounts for coincidence detection across multiple afferents and provides a biologically-plausible principle that generalizes the Bienenstock-Cooper-Munro (BCM) rule [2] for spiking neurons. In order to focus on the first spike latencies of the mechanoreceptor signals, we adapt the learning rule developed by Toyoizumi *et al.* 2005 [20] to very short transient stimuli, and we apply it to maximize the information transfer at the level of the CN neural population. See Supporting Material Sec. A.2 for details on the learning rule. The weights of mechanoreceptor-CN synapses are initialized randomly between 0 and 1 according to a uniform distribution. The training phase consists of 200 presentations of the sequence of 81 stimuli.

## 2.3 Metrical information transfer measure

An information-theoretical approach is employed to assess the efficiency of the haptic discrimination process. Classical literature solutions based on Shannon's mutual information (MI) [19] consist of using either a binning procedure (which reduces the response space and relaxes the temporal constraint) or a clustering method (e.g. k-nearest neighbors based on spike-train metrics) coupled to a confusion matrix to estimate a lower bound on MI. Yet, none of these techniques allows the information transmission to be assessed by taking into full account the metrics of the spike response space. Furthermore, a decoding scheme accounting for precise temporal discrimination while maintaining the combinatorial properties of the output space within suitable boundaries – even in the presence of hundreds of CN spike trains – is needed.

A novel definition of entropy is set forth to provide a suitable measure for the encoding/decoding of spiking signals, and to quantify the information transmission in the presence of large populations of

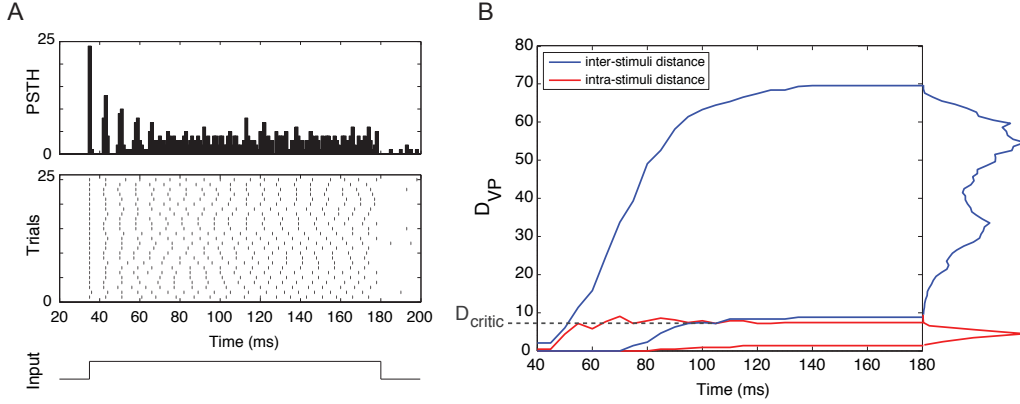

Figure 2: (A) Example of discharge patterns of a model CN neuron evoked by a constant depolarizing current (bottom). Responses are shown as a raster plot of spike times during 25 trials (center), and as the corresponding PSTH (top). (B) Example of intra- and inter-stimulus distances $D_{VP}$ (red and blue curves, respectively) over time for a VP cost parameter $C_{VP} = 0.15$. The optimal discrimination condition is met after about 110 ms, when the distribution of intra- and inter-stimulus distances (right plot) stop overlapping. Fig. S2 in the Supporting Material shows an example of two distance distributions that never become disjoint (i.e. perfect discrimination never occurs).

spike trains with a 1 ms temporal precision. The following definition of entropy is taken:

$$H^*(R) = -\sum_{r \in R} \frac{1}{|R|} \log \sum_{r' \in R} \frac{<r|r'>}{|R|} \tag{1}$$

where $R$ is the set of responses elicited by all the stimuli, $|R|$ is the cardinal of $R$, and $<r|r'>$ is a similarity measure between any two responses $r$ and $r'$. The similarity measure $<r|r'>$ depends on Victor-Purpura (VP) spike train metrics [23] (see below).

It is worth noting that, in contrast to the Shannon definition of entropy, in which the sum is over different response clusters, here the sum is over all the $|R|$ responses, no matter if they are identical or different (i.e. cluster-less entropy definition). Also, the similarity measure $<r|r'>$ allows the computation of the probability of getting a given response (i.e. $p(r|s)$) to be avoided, which usually implies to group responses in clusters. These aspects make $H^*(R)$ suitable to take into account the metric properties of the responses. Notice that if the similarity measure were defined as $<r|r'> = \delta(r, r')$ (with $\delta$ being the Dirac function), then $H^*(R)$ would be exactly the same as the Shannon entropy.

The conditional entropy is then taken as:

$$H^*(R|S) = \sum_{s \in S} p(s) H^*(R|s) = -\sum_{s \in S} p(s) \sum_{r \in R_s} \frac{1}{|R_s|} \log \sum_{r' \in R_s} \frac{<r|r'>}{|R_s|} \tag{2}$$

where $R_s$ is the set of responses elicited by the stimulus $s$.

Finally, the metrical information measure is given by:

$$I^*(R; S) = H^*(R) - H^*(R|S) \tag{3}$$

The similarity measure $<r|r'>$ is defined as a function of the VP distance $D_{VP}(r, r')$ between two population responses $r$ and $r'$. The distance $D_{VP}(r, r')$ depends on the VP cost parameter $C_{VP}$ [23], which determines the time scale of the analysis by regulating the influence of spike timing *vs.* spike count when calculating the distance between $r$ and $r'$.

There is an infinite number of ways to obtain a scalar product from a distance. We take a very simple one, defined as:

$$<r|r'> = 1 \iff D_{VP}(r, r') < D_{\text{critic}} \tag{4}$$

where the critical distance $D_{\text{critic}}$ is a free parameter. According to Eq. 4, whenever $D_{VP}(r, r') < D_{\text{critic}}$ the responses $r, r'$ are considered to be identical, otherwise they are classified as different. If $D_{\text{critic}} = 0$ one recovers the Shannon entropy from Eq. 1.

In order to determine the optimal value for $D_{\text{critic}}$, we consider two sets of VP distances:

- the *intra*-stimulus distances $D_{VP}(r(s), r'(s))$ between responses $r, r'$ elicited by the same stimulus $s$;
- the *inter*-stimulus distances $D_{VP}(r(s), r'(s''))$ between responses $r, r'$ elicited by two different stimuli $s, s''$.

Then, we compute the minimum and maximum intra-stimulus distances as well as the minimum and maximum inter-stimulus distances. The optimal coding condition, corresponding to maximum $I^*(R; S)$ and zero $H^*(R|S)$, occurs when the maximum intra-stimulus distance becomes smaller than the minimum inter-stimulus distance.

In the case of spike train neurotransmission, the relationship between intra- and inter-stimulus distance distributions tends to evolve over time, as the input spike wave across multiple afferents flows in. Fig. 2B shows an example of intra- and inter-stimulus distance distributions evolving over time. The two distributions separate from each other after about 110 ms. The critical parameter $D_{\text{critic}}$ can then be taken as the distance at which the maximum intra-stimulus distance becomes smaller than the minimum inter-stimulus distance (dashed line in Fig. 2B). The time at which the critical distance $D_{\text{critic}}$ can be determined (i.e. the time at which the two distributions stop overlapping) indicates when the perfect discrimination condition is reached (i.e. maximum $I^*(R; S)$ and zero $H^*(R|S)$).

To summarize, perfect discrimination calls upon the following rule:

- if all intra-stimulus distances are smaller than the critical distance $D_{\text{critic}}$, then all the responses elicited by any stimulus are considered identical. The conditional entropy $H^*(R|S)$ is therefore nil.
- if all inter-stimulus distances are greater than $D_{\text{critic}}$, then two responses elicited by two different stimuli are always discriminated. The information $I^*(R; S)$ is therefore maximum.

As aforementioned, the critical distance $D_{\text{critic}}$ is interdependent on the VP cost parameter $C_{VP}$ [23]. We define the optimum VP cost $C_{VP}^*$ as the one that leads to earliest perfect discrimination (in the example of Fig. 2B, a cost $C_{VP} = 0.15$ leads to perfect discrimination after 110 ms).

## 3  Results

### 3.1  Decoding of spiking haptic signals upstream from the cuneate nucleus

First, we validate the information theoretical analysis described above to decode a limited set of microneurography data upstream from the CN network [18]. Only the 5 force directions (ulnar, radial, distal, proximal, normal) are considered as variable primary features [9]. Each of the 5 stimuli is presented 100 times, and the VP distances $D_{VP}$ are computed across the population of 42 mechanoreceptor afferents. Fig. 3A shows that the critical distance $D_{critic} = 8$ can be set 72 ms after the stimulus onset. As shown in Fig. 3B, that ensures that the perfect discrimination condition is met within 30 ms of the first mechanoreceptor discharge. Fig. 3C displays two samples of distance matrices indicating how the input spike waves across the 42 mechanoreceptor afferents are clustered by the decoding system over time. Before the occurrence of the perfect discrimination condition (left matrix) different stimuli can have relatively small distances (e.g. P and N force directions), which means that some interferences are affecting the decoding process. After 72 ms (right matrix), all the initially overlapping contexts become pulled apart, which removes all interferences across inputs and leads to a $100\%$ accuracy in the discrimination process.

### 3.2  Optimal haptic context separation downstream from the cuneate nucleus

Second, the entire set of microneurography recordings (81 stimuli) is employed to analyze the information transmission properties of a network of 50 CN neurons in the presence of synaptic plasticity

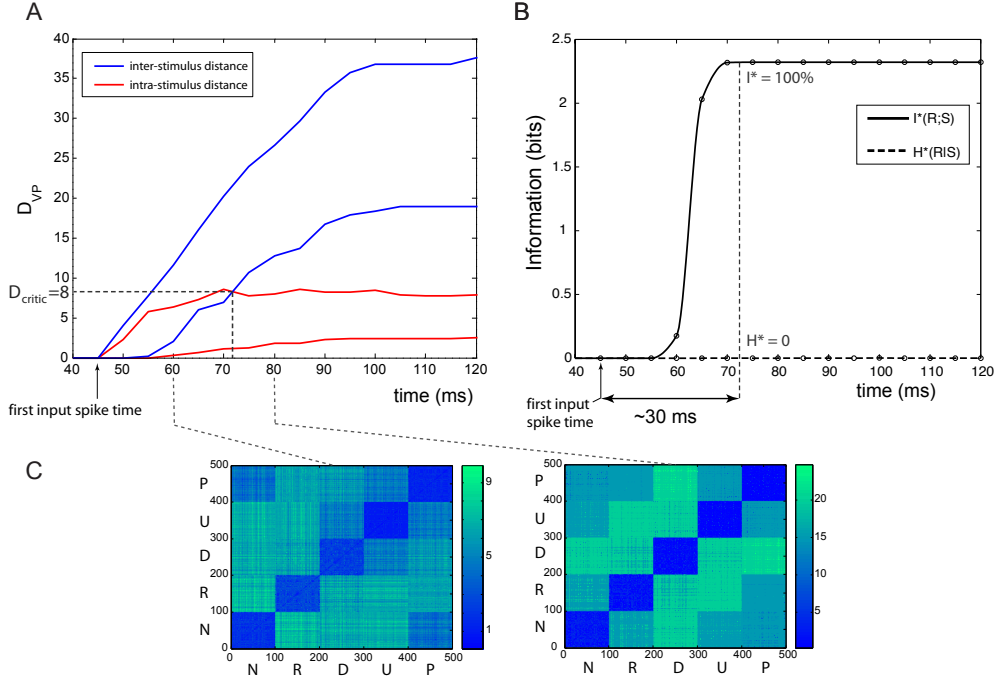

Figure 3: Discrimination capacity upstream from the CN for a set of 5 stimuli (obtained by varying the orientation parameter only) presented 100 times each. (A) Intra- and inter-stimulus distances over time for a VP cost parameter $C_{VP} = 0.15$. The perfect discrimination condition is met 72 ms after the stimulus onset and 30 ms after the arrival of the first spike. (B) Metrical information and conditional entropy obtained with $D_{critic} = 8$. (C) Distance matrices before and after the occurrence of perfect discrimination.

(i.e. LTP/LTD based on the learning rule detailed in Sec. A.2). To compute $I^*(R; S)$, the VP distances $D_{VP}(r, r')$ between any two CN population responses $r, r'$ are considered. Again, the distance $D_{critic}$ is used to identify the perfect discrimination condition, and the VP cost parameter $C^*_{VP} = 0.1$ yielding the fastest perfect discrimination is selected. Fig. 4A shows that the CN population achieves optimal context separation within 35 ms of the arrival of the first afferent spikes.

Selecting the optimal value of the critical distance, as done for Fig. 4A, corresponds to the situation in which a readout system downstream from the CN would need a complete separation of haptic percepts (e.g. for highly precise feature recognition). Relaxing this optimality constraint (e.g. to the extent of very rapid, though less precise, reactions) can further speed up the discrimination process. For instance, Fig. 4B indicates that setting $D_{critic}$ to a suboptimal value would lead to a partial discrimination condition in which 80% of the maximum $I^*(R; S)$ (with *non-zero* $H^*(R|S)$) can be achieved within 15 ms of the arrival of the first pre-synaptic spike.

Figs. 4C-D illustrate the distributions of intra- and inter-stimulus distances 100 ms after stimulus onset before and after learning. It is shown that while the distributions are well-separated after learning, they are still largely overlapping before training (implying the impossibility of perfect discrimination). It is also interesting to note that after (resp. before) learning the CN fired on average n=217 (resp. 39) spikes, and that the maximum intra-stimulus distance was about $D^{max}_{VP}$=14 (resp. 45). The average uncertainty on the timing of a single spike can be expressed by $\Delta t = D^{max}_{VP} / C_{VP}n$. Since $C_{VP} = 0.1$, $\Delta t = 0.6$ ms after learning and $\sim 12$ ms before. This shows that the plasticity rule helped to reduce the jitter on CN spikes, thus reducing the metrical conditional entropy compared to the pre-learning condition.

Fig. 4E suggests that the plasticity rule leads to stable weight distributions that are invariant with respect to initial random conditions (uniform distribution between $[0, 1]$). After learning, the synaptic

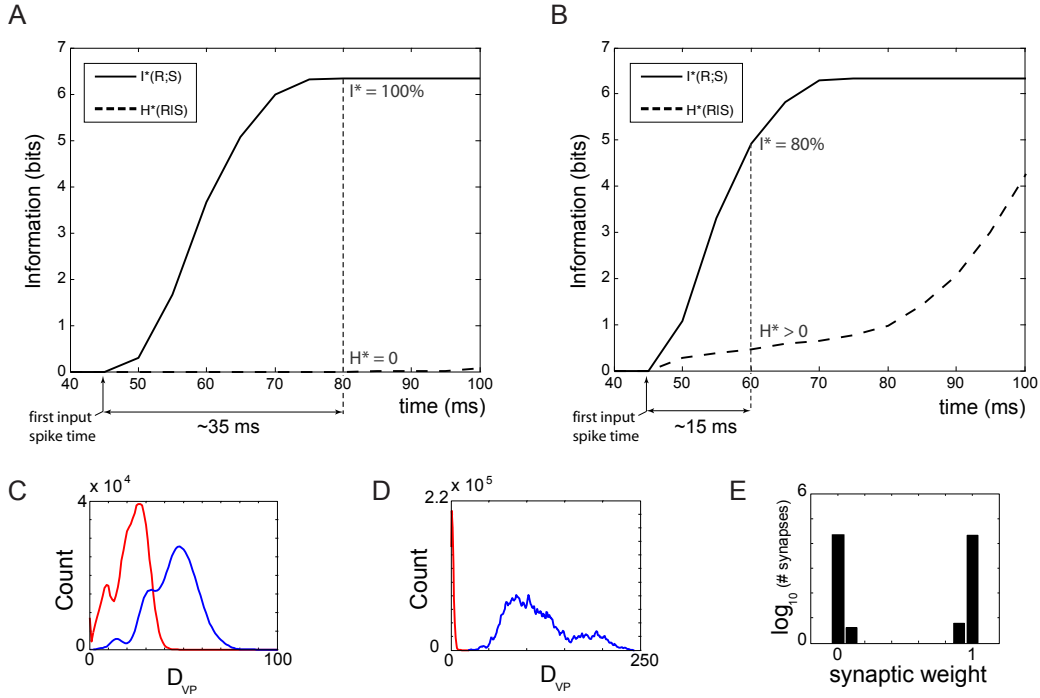

Figure 4: Information $I^*(R;S)$ and conditional entropy $H^*(R|S)$ over time. The CN population consists of 50 cells. The 81 tactile stimuli are presented 100 times each. (A) Optimal discrimination is reached 35 ms after the first afferent spike. (B) If the perfect discrimination constraint is relaxed by reducing the critical distance, then the system can perform partial discrimination –i.e 80% of maximum $I^*(R;S)$ and non-zero $H^*(R|S)$– already within 15 ms of the first spike time. (C-D) Distributions of intra- and inter-stimulus distances (computed 100 ms after stimulus onset) before and after training, respectively. (E) Distribution of CN synaptic weights after learning. In this example, a network of 10000 cuneate neurons has been trained.

efficacies of the mechanoreceptor-to-CN projections converge towards a bimodal distribution with one peak close to zero and the other peak close to the maximum weight.

Finally, Sec. A.3 and Fig. S3 report some supplementary results obtained by using a classical STDP rule [1, 15] –rather than the learning rule described in Secs. 2.2 and A.2– to train the CN network.

### 3.3 How does the size of the cuneate nucleus network influence discrimination?

An additional analysis was performed to study the relationship between the size of the CN population and the optimality of the encoding/decoding process. This analysis reveals that a lower bound on the number of CN neurons exists in order to perform optimal (i.e. both very rapid and reliable) discrimination of the 81 microneurography spike trains. As shown in Fig. 5, the perfect discrimination condition cannot be met with a population of less than 50 CN neurons. This result corroborates the hypothesis that a spatiotemporal population code is a necessary condition for performing effective context separation of complex spiking signals [3, 6]. By increasing the number of neurons, the discrimination becomes faster and saturates at 72 ms (which corresponds to the time at which the first spike from the slowest volley of pulses arrives at the CN). It is also shown that the number of spikes emitted on average by CN cells under the optimal discrimination condition decreases from 2.1 to 1.3 with the size of the CN population, supporting the idea that one spike per neuron is enough to convey a significant amount of information.

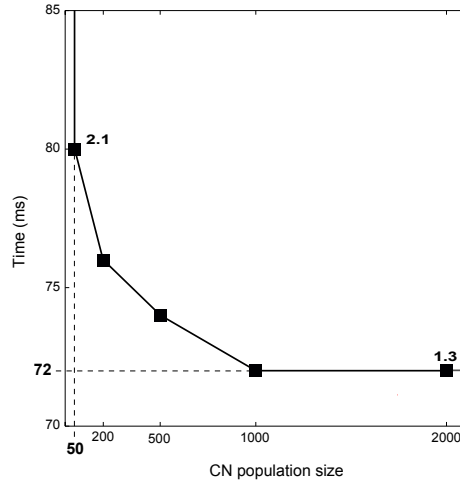

Figure 5: Time necessary to perfectly discriminate the entire set of 81 stimuli as a function of the size of the CN population. Each stimulus is presented 100 times. The numbers of spikes emitted on average by each CN neuron when optimal discrimination occurs are also indicated in the diagram.

## 4 Discussion

This study focuses on how a population of $2^{nd}$ order somatosensory neurons in the cuneate nucleus (CN) can encode incoming spike trains –obtained via microneurography recordings in humans– by separating them in an abstract metrical space. The main contribution is the prediction concerning a significant role of the CN in conveying optimal contextual accounts of peripheral tactile inputs to downstream structures of the CNS.

It is shown that an encoding/decoding mechanism based on relative spike timing can account for rapid and reliable transmission of tactile information at the level of the CN. In addition, it is emphasized that the variability of the CN conditioned responses to tactile stimuli constitutes a fundamental measure when examining neurotransmission at this stage of the ascending somatosensory pathway. More generally, the number of responses elicited by a stimulus is a critical issue when information has to be transferred through multiple synaptic relays. If a single stimulus can possibly elicit millions of different responses on a neural layer, how can this plethora of data be effectively decoded by downstream networks? Thus, neural information processing requires encoding mechanisms capable of producing as few responses as possible to a given stimulus while keeping these responses different between stimuli.

A corollary contribution of this work consists in putting forth a novel definition of entropy, $H^*(R)$, to assess neurotransmission in the presence of large spike train spaces and with high temporal precision. An information theoretical analysis –based on this novel definition of entropy– is used to measure the ability of CN network to perform haptic context discrimination. The optimality condition corresponds to maximum information $I^*(R; S)$ and (simultaneously) minimum conditional entropy $H^*(R|S)$ (which quantifies the variability of the CN conditioned responses).

Finally, the proposed information theoretical measure accounts for the metrical properties of the response space explicitly and estimates the optimality of the encoding/decoding process based on its context separation capability (which minimizes destructive interference over learning and maximizes memory capacity). The method does not call upon an *a priori* decoding analysis to build predefined response clusters (e.g. as the confusion matrix method does to compute conditional probabilities and then Shannon MI). Rather, the evaluation of the clustering process is embedded in the entropy measure and, when the condition of optimal discrimination is reached, the existence of well-defined clusters is ensured.

**Acknowledgments**. Granted by the EC Project SENSOPAC, IST-027819-IP.

# References

[1] G. Bi and M. Poo. Distributed synaptic modification in neural networks induced by patterned stimulation. *Nature*, 401:792–796, 1999.

[2] E. Bienenstock, L. Cooper, and P. Munro. Theory for the development of neuron selectivity: orientation specificity and binocular interaction in visual cortex. *J Neurosci*, 2:32–48, 1982.

[3] S. Furukawa, L. Xu, and J.C. Middlebrooks. Coding of sound-source location by ensembles of cortical neurons. *J Neurosci*, 20:1216–1228, 2000.

[4] T.J. Gawne, T.W. Kjaer, and B.J. Richmond. Latency: another potential code for feature-binding in the striate cortex. *J Neurophysiol*, 76:1356–1360, 1996.

[5] W. Gerstner and W. Kistler. *Spiking Neuron Models*. Cambridge University Press, 2002.

[6] T. Gollisch and M. Meister. Rapid neural coding in the retina with relative spike latencies. *Science*, 319:1108–1111, 2008.

[7] P. Heil. First-spike latency of auditory neurons revisited. *Curr Opin Neurobiol*, 14:461–467, 2004.

[8] J.J. Hopfield. Pattern recognition computation using action potential timing for stimulus representation. *Nature*, 376:33–36, 1995.

[9] R.S. Johansson and I. Birznieks. First spikes in ensembles of human tactile afferents code complex spatial fingertip events. *Nat Neurosci*, 7:170 – 177, 2004.

[10] R.S. Johansson and J.R. Flanagan. Coding and use of tactile signals from the fingertips in object manipulation tasks. *Nat Rev Neurosci*, 10:345–359, 2009.

[11] R.S. Johansson and A. Vallbo. Tactile sensory coding in the glabrous skin of the human hand. *Trends Neurosci*, 6:27–32, 1983.

[12] E. Jones. Cortical and subcortical contributions to activity-dependent plasticity in primate somatosensory cortex. *Annu Rev Neurosci*, 23:1–37, 2000.

[13] P. Koenig, A.K. Engel, and W. Singer. Integrator or coincidence detector? The role of the cortical neuron revisited. *Trends Neurosci*, 19:130–137, 1996.

[14] Z.F. Mainen and T.J. Sejnowski. Reliability of spike timing in neocortical neurons. *Science*, 268:1503–1506, 1995.

[15] H. Markram, J. Luebke, M. Frotscher, and B. Sakmann. Regulation of synaptic efficacy by coincidence of postsynaptic APs and EPSPs. *Science*, 275:213–215, 1997.

[16] I. Nelken, G. Chechik, T.D. Mrsic-Flogel, A.J. King, and J.W. Schnupp. Encoding stimulus information by spike numbers and mean response time in primary auditory cortex. *J Comput Neurosci*, 19:199–221, 2005.

[17] S. Panzeri, R.S. Petersen, S.R Schultz, M. Lebedev, and M.E. Diamond. The role of spike timing in the coding of stimulus location in rat somatosensory cortex. *Neuron*, 29:769–777, 2001.

[18] H.P. Saal, S. Vijayakumar, and R.S. Johansson. Information about complex fingertip parameters in individual human tactile afferent neurons. *J Neurosci*, 29:8022–8031, 2009.

[19] C.E. Shannon. A mathematical theory of communication. *Bell Sys Tech J*, 27:379–423, 1948.

[20] T. Toyoizumi, J.-P. Pfister, K. Aihara, and W. Gerstner. Generalized Bienenstock-Cooper-Munro rule for spiking neurons that maximizes information transmission. *Proc Natl Acad Sci U S A*, 102(14):5239–5244, 2005.

[21] A. Vallbo and R.S. Johansson. Properties of cutaneous mechanoreceptors in the human hand related to touch sensation. *Hum Neurobiol*, 3:3–14, 1984.

[22] R. VanRullen, R. Guyonneau, and S.J. Thorpe. Spike time make sense. *Trends Neurosci*, 28:1–4, 2005.

[23] J.D. Victor and K.P. Purpura. Nature and precision of temporal coding in visual cortex: a metric-space analysis. *J Neurophysiol*, Vol 76:1310–1326, 1996.

